# Transelliptical Component Analysis

**Fang Han**
Department of Biostatistics
Johns Hopkins University
Baltimore, MD 21210
fhan@jhsph.edu

**Han Liu**
Department of Operations Research
and Financial Engineering
Princeton University, NJ 08544
hanliu@princeton.edu

## Abstract

We propose a high dimensional semiparametric scale-invariant principle component analysis, named TCA, by utilize the natural connection between the elliptical distribution family and the principal component analysis. Elliptical distribution family includes many well-known multivariate distributions like multivariate Gaussian, t and logistic and it is extended to the meta-elliptical by Fang et.al (2002) using the copula techniques. In this paper we extend the meta-elliptical distribution family to a even larger family, called *transelliptical*. We prove that TCA can obtain a near-optimal $s\sqrt{\log d/n}$ estimation consistency rate in recovering the leading eigenvector of the latent generalized correlation matrix under the transelliptical distribution family, even if the distributions are very heavy-tailed, have infinite second moments, do not have densities and possess arbitrarily continuous marginal distributions. A feature selection result with explicit rate is also provided. TCA is further implemented in both numerical simulations and large-scale stock data to illustrate its empirical usefulness. Both theories and experiments confirm that TCA can achieve model flexibility, estimation accuracy and robustness at almost no cost.

## 1 Introduction

Given $x_1, \ldots, x_n \in \mathbb{R}^d$ as $n$ i.i.d realizations of a random vector $X \in \mathbb{R}^d$ with population covariance matrix $\Sigma$ and correlation matrix $\Sigma^0$, the Principal Component Analysis (PCA) aims at recovering the top $m$ leading eigenvectors $u_1, \ldots, u_m$ of $\Sigma$. In practice, $\Sigma$ is unknown and the top $m$ leading eigenvectors $\widehat{u}_1, \ldots, \widehat{u}_m$ of the Pearson sample covariance matrix are obtained as the estimators. However, because the PCA is well-known to be scale-variant, meaning that changing the measurement scale of variables will make the estimators different, the PCA conducted on the sample correlation matrix is also regular in literatures [2]. It aims at recovering the top $m$ leading eigenvectors $\theta_1, \ldots, \theta_m$ of $\Sigma^0$ using the top $m$ leading eigenvectors $\widehat{\theta}_1, \ldots, \widehat{\theta}_m$ of the Pearson sample correlation matrix. Because $\Sigma^0$ is scale-invariant, we call the PCA aiming at recovering the eigenvectors of $\Sigma^0$ the scale-invariant PCA.

In high dimensional settings, when $d$ scales with $n$, it has been discussed in [14] that $\widehat{u}_1$ and $\widehat{\theta}_1$ are generally not consistent estimators of $u_1$ and $\theta_1$. For any two vectors $v_1, v_2 \in \mathbb{R}^d$, denote the angle between $v_1$ and $v_2$ by $\angle(v_1, v_2)$. [14] proved that $\angle(u_1, \widehat{u}_1)$ and $\angle(\theta_1, \widehat{\theta}_1)$ do not converge to zero. Therefore, it is commonly assumed that $\theta_1 = (\theta_{11}, \ldots, \theta_{1d})^T$ is sparse, meaning that $\mathrm{card}(\mathrm{supp}(\theta_1)) := \mathrm{card}(\{\theta_{1j} : \theta_{1j} \neq 0\}) = s < n$. This results in a variety of sparse PCA procedures. Here we note that $\mathrm{supp}(u_j) = \mathrm{supp}(\theta_j)$, for $j = 1, \ldots, d$.

The *elliptical distributions* are of special interest in Principal Component Analysis. The study of elliptical distributions and their extensions have been launched in statistics recently by [4]. The elliptical distributions can be characterized by their stochastic representations [5]. A random vector $Z = (Z_1, \ldots, Z_d)^T$ is said to follow an elliptical distribution or be elliptically distributed with parameters $\mu, \Sigma \succeq 0$, and $\mathrm{rank}(\Sigma) = q$, if it admits the stochastic representation: $Z = \mu + \xi A U$, where $\mu \in \mathbb{R}^d$, $\xi \in \mathbb{R}$ and $U \in \mathbb{R}^q$ are independent random variables, $\xi \geq 0$, $U$ is uniformly distributed on the unit sphere in $\mathbb{R}^q$, and $A \in \mathbb{R}^{d \times q}$ is a fixed matrix such that $AA^T = \Sigma$. We call

$\xi$ the *generating variable*. The density of $Z$ does not necessarily exist. Elliptical distribution family includes a variety of famous multivariate distributions: multivariate Gaussian, multivariate Cauchy, Student's t, logistic, Kotz, symmetric Pearson type-II and type-VII distributions. We refer to [3, 5] and [4] for more details.

[4] introduce the term *meta-elliptical distribution* in extending the continuous elliptical distributions whose densities exist to a wider class of distributions with densities existing. The construction of the meta-elliptical distributions is based on the *copula* technique and it was initially introduced by [25]. In particular, when the latent elliptical distribution is the multivariate Gaussian, we have the *meta-Gaussian* or the *nonparanormal* distributions introduced by [16] and [19].

The elliptical distribution is of special interest in Principal Component Analysis (PCA). It has been shown in a variety of literatures [27, 11, 22, 12, 24] that the PCA conducted on elliptical distributions shares a number of good properties enjoyed by the PCA conducted on the Gaussian distribution. In particular, [11] show that with regard to a range of hypothesis relevant to PCA, tests based on a multivariate Gaussian assumption have the identical power for all elliptical distributions even without second moments. We will utilize this connection to construct a new model in this paper.

In this paper, a new high dimensional scale-invariant principle component analysis approach is proposed, named $\underline{T}$ranselliptical $\underline{C}$omponent $\underline{A}$nalysis (TCA). Firstly, to achieve both the estimation accuracy and model flexibility, we build the model of TCA on the *transelliptical distributions*. A random vector $X = (X_1, \ldots, X_d)^T$ is said to follow a transelliptical distribution if there exists a set of univariate strictly monotone functions $f = \{f_j\}_{j=1}^d$ such that $f(X) := (f_1(X_1), \ldots, f_d(X_d))^T$ follows a continuous elliptical distribution with parameters $\mu = 0$ and $\Sigma^0 = [\Sigma_{jk}^0] \succeq 0$. Here $\mathrm{diag}(\Sigma^0) = \mathbf{1}$. Transelliptical distributions do not necessarily possess densities and are strict extensions to the meta-elliptical distributions defined in [4]. TCA aims at recovering the top $m$ leading eigenvectors $\theta_1, \ldots, \theta_m$ of $\Sigma^0$.

Secondly, to estimate $\Sigma^0$ robustly and efficiently, instead of estimating the transformation functions $\{\widehat{f}_j\}_{j=1}^d$ of $\{f_j\}_{j=1}^d$ as [19] did, realizing that $\{f_j\}_{j=1}^d$ preserve the ranks of the data, we utilize the nonparametric rank-based correlation coefficient estimator, Kendall's tau, to estimate $\Sigma^0$. We prove that even though the generating variable $\xi$ is changing and marginal distributions are arbitrarily continuous, Kendall's tau correlation matrix approximates $\Sigma^0$ in a parametric rate $O_P(\sqrt{\log d/n})$. This key observation makes Kendall's tau a better estimator than Pearson sample correlation matrix with regard to a much larger distribution family than the Gaussian.

Thirdly, in terms of methodology and theory, we analyze the general case that $X$ follows a transelliptical distribution and $\theta_1$ is sparse. Here $\theta_1$ is the leading eigenvector of $\Sigma^0$. We obtain the TCA estimator $\widetilde{\theta}_1^*$ of $\theta_1$ utilizing the Kendall's tau correlation matrix. We prove that the TCA can obtain a fast convergence rate in terms of parameter estimation and is of the rate $\sin \angle(\theta_1, \widetilde{\theta}_\infty) = O_P(s\sqrt{\log d/n})$, where $\widetilde{\theta}_\infty$ is the estimator TCA obtains. A feature selection consistency result with explicit rate is also provided.

## 2  Background

We start with notations: Let $M = [M_{jk}] \in \mathbb{R}^{d \times d}$ and $v = (v_1, ..., v_d)^T \in \mathbb{R}^d$. Let $v$'s subvector with entries indexed by $I$ be denoted by $v_I$, $M$'s submatrix with rows indexed by $I$ and columns indexed by $J$ be denoted by $M_{IJ}$. Let $M_{I\cdot}$ and $M_{\cdot J}$ be the submatrix of $M$ with rows in $I$ and all columns, and the submatrix of $M$ with columns in $J$ and all rows. For $0 < q < \infty$, we define the $\ell_0$, $\ell_q$ and $\ell_\infty$ vector norm as

$$\|v\|_0 := \mathrm{card}(\mathrm{supp}(v)), \quad \|v\|_q := (\sum_{i=1}^d |v_i|^q)^{1/q} \text{ and } \|v\|_\infty := \max_{1 \le i \le d} |v_i|.$$

We define the matrix $\ell_{\max}$ norm as the elementwise maximum value: $\|M\|_{\max} := \max\{|M_{ij}|\}$ and the $\ell_\infty$ norm as $\|M\|_\infty := \max_{1 \le i \le m} \sum_{j=1}^n |M_{ij}|$. Let $\Lambda_j(M)$ be the toppest $j-$th eigenvalue of M. In special, $\Lambda_{\min}(M) := \Lambda_d(M)$ and $\Lambda_{\max}(M) := \Lambda_1(M)$ are the smallest and largest eigenvalues of $M$. The vectorized matrix of $M$, denoted by $\mathrm{vec}(M)$, is defined as: $\mathrm{vec}(M) := (M_{\cdot 1}^T, \ldots, M_{\cdot d}^T)^T$. Let $\mathbb{S}^{d-1} := \{v \in \mathbb{R}^d : \|v\|_2 = 1\}$ be the $d$-dimensional unit sphere. The sign $=^d$ denotes that the two sides of the equality have the same distributions. For any two vectors $a, b \in \mathbb{R}^d$ and any two squared matrices $A, B \in \mathbb{R}^{d \times d}$, denote the inner product of $a$ and $b$, $A$ and

$B$ by

$$\langle a, b \rangle := a^T b \quad \text{and} \quad \langle A, B \rangle := \text{Tr}(A^T B).$$

## 2.1 Elliptical and Transelliptical Distributions

This section is devoted to a brief discussion of elliptical and transelliptical distributions. In the sequel, to be clear, a random vector $X = (X_1, \ldots, X_d)^T$ is said to be *continuous* if the marginal distribution functions are all continuous.

### 2.1.1 Elliptical Distributions

In this section we shall firstly provide a definition of the elliptical distributions following [5].

**Definition 2.1.** Given $\mu \in \mathbb{R}^d$ and $\Sigma \in \mathbb{R}^{d \times d}$, where $\text{rank}(\Sigma) = q \leq d$, a random vector $Z = (Z_1, \ldots, Z_d)^T$ is said to have an elliptical distribution or is elliptically distributed with parameters $\mu$ and $\Sigma$, if and only if $Z$ has a stochastic representation: $Z =^d \mu + \xi A U$, where $\mu \in \mathbb{R}^d$, $A \in \mathbb{R}^{d \times q}$, $AA^T = \Sigma$, $\xi \geq 0$ is a random variable independent of $U$, $U \in \mathbb{S}^{q-1}$ is uniformly distributed in the unit sphere in $\mathbb{R}^q$. In this setting we denote by $Z \sim EC_d(\mu, \Sigma, \xi)$.

A random variable in $\mathbb{R}$ with continuous marginal distribution function does not necessarily possess density. A well-known set of examples is the *cantor distribution*, whose support set is the cantor set. We refer to [7] for more discussions on this phenomenon. $\Sigma$ is symmetric and positive semi-definite, but not necessarily to be positive definite.

**Proposition 2.1.** *A random vector $Z = (Z_1, \ldots, Z_d)^T$ has the stochastic representation $Z \sim EC_d(\mu, \Sigma, \xi)$, if and only if $Z$ has the characteristic function $\exp(it'\mu)\phi(t'\Sigma t)$, where $\phi$ is a properly-defined characteristic function. We denote by $X \sim EC_d(\mu, \Sigma, \phi)$. If $\xi$ is absolutely continuous and $\Sigma$ is non-singular, then the density of $Z$ exists and is of the form: $p_Z(z) = |\Sigma|^{-1/2} g\left((z - \mu)^T \Sigma^{-1}(z - \mu)\right)$, where $g : [0, \infty) \to [0, \infty)$. We denote by $Z \sim EC_d(\mu, \Sigma, g)$.*

A proof can be found in page 42 of [5]. When the density exists, $\xi$, $\phi$ and $g$ are uniquely determined by one of the other. The relationship among $\xi$, $\phi$ and $g$ are described in Theorem 2.2 and Theorem 2.9 of [5]. The next proposition states that $\Sigma$, $\phi$, $\xi$ and $A$ are not unique.

**Proposition 2.2 (Theorem 2.15 of [5]).** *(i) If $Z = \mu + \xi A U$ and $Z = \mu^* + \xi^* A^* U^*$, where $A \in \mathbb{R}^{d \times q}$ and $A^* \in \mathbb{R}^{d \times q}$, $Z$ is continuous, then there exists a constant $c > 0$ such that $\mu^* = \mu$, $A^* A^{*T} = cAA^T$, $\xi^* = c^{-1/2}\xi$. (ii) If $Z \sim EC_d(\mu, \Sigma, \phi)$ and $Z \sim EC_d(\mu^*, \Sigma^*, \phi^*)$, $Z$ is continuous, then there exists a constant $c > 0$ such that $\mu^* = \mu$, $\Sigma^* = c\Sigma$, $\phi^*(\cdot) = \phi(c^{-1}\cdot)$.*

The next proposition discusses the cases where $(\mu, \Sigma, \xi)$ is identifiable for $Z$.

**Proposition 2.3.** *If $Z \sim EC_d(\mu, \Sigma, \xi)$ is continuous with $\text{rank}(\Sigma) = q$, then (1) $\mathbb{P}(\xi = 0) = 0$; (2) $\Sigma_{ii} > 0$ for $i \in \{1, \ldots, d\}$; (3) $(\mu, \Sigma, \xi)$ is identifiable for $Z$ under the constraint that $\max(\text{diag}(\Sigma)) = 1$.*

We define $\Sigma^0 = [\Sigma^0_{jk}]$ with $\Sigma^0_{jk} = \Sigma_{jk}/\sqrt{\Sigma_{jj}\Sigma_{kk}}$ to be the *generalized correlation matrix* of $Z$. $\Sigma^0$ is the correlation matrix of $Z$ when $Z$'s second moment exists and still reflects the rank dependency even when $Z$ has infinite second moment [13].

### 2.1.2 Transelliptical Distributions

To extend the elliptical distribution, we firstly define two sets of symmetric matrices: $\mathcal{R}_d^+ = \{\Sigma \in \mathbb{R}^{d \times d} : \Sigma^T = \Sigma, \text{diag}(\Sigma) = \mathbf{1}, \Sigma \succ 0\}$; $\mathcal{R}_d = \{\Sigma \in \mathbb{R}^{d \times d} : \Sigma^T = \Sigma, \text{diag}(\Sigma) = \mathbf{1}, \Sigma \succeq 0\}$.

**Definition 2.2.** A random vector $X = (X_1, \ldots, X_d)^T$ with continuous marginal distribution functions $F_1, \ldots, F_d$ and density existing is said to follow a meta-elliptical distribution if and only if there exists a continuous elliptically distributed random vector $Z \sim EC_d(0, \Sigma^0, g)$ with the marginal distribution function $Q_g$ and $\Sigma^0 \in \mathcal{R}_d^+$, such that $(Q_g^{-1}(F_1(X_1)), \ldots, Q_g^{-1}(F_d(X_d)))^T =^d Z$.

In this paper, we generalize the meta-elliptical distribution family to a broader class, named the transelliptical. The transelliptical distributions do not assume that densities exist for both $X$ and $Z$ and are therefore strict extensions to meta-elliptical distributions.

**Definition 2.3.** A random vector $X = (X_1, \ldots, X_d)^T$ is said to follow a transelliptical distribution if and only if there exists a set of strictly monotone functions $f = \{f_j\}_{j=1}^d$ and a latent continuous elliptically distributed random vector $Z \sim EC_d(0, \Sigma^0, \xi)$ with $\Sigma^0 \in \mathcal{R}_d$, such that $(f_1(X_1), \ldots, f_d(X_d))^T =^d Z$. We call such $X \sim TE_d(\Sigma^0, \xi; f_1, \ldots, f_d)$ and $\Sigma^0$ the latent generalized correlation matrix.

**Proposition 2.4.** *If $X$ follows a meta-elliptical distribution, in other words, $X$ possesses density and has continuous marginal distributions $F_1, \ldots, F_d$ of $X$ and a continuous random vector $Z \sim EC_d(0, \Sigma^0, g)$ such that $(Q_g^{-1}(F_1(X_1)), \ldots, Q_g^{-1}(F_d(X_d)))^T =^d Z$, then we have $X \sim TE_d(\Sigma^0, \xi; Q_g^{-1}(F_1), \ldots, Q_g^{-1}(F_d))$.*

To be more clear, the transelliptical distribution family is strictly larger than the meta-elliptical distribution family in three senses: (i) the generating variable $\xi$ of the latent elliptical distribution is not necessarily absolute continuous in transelliptical distributions; (ii) the parameter $\Sigma^0$ is strictly enlarged from $\mathcal{R}_d^+$ to $\mathcal{R}_d$; (iii) the marginal distributions of $X$ do not necessarily possess densities.

The term *meta-Gaussian* (or the nonparanormal) is introduced by [16, 19]. The term *meta-elliptical copula* is introduced in [6]. This is actually an alternative definition of the meta-elliptical distribution. The term *elliptical copula* is introduced in [18]. In summary,

transelliptical $\supset$ meta-elliptical = meta-elliptical copula $\supset$ elliptical* $\supset$ elliptical copula,

transelliptical $\supset$ meta-Gaussian = nonparanormal.

Here elliptical* represents the elliptical distributions which are continuous and possess densities.

### 2.2 Latent Correlation Matrix Estimation for Transelliptical Distributions

We firstly study the correlation and covariance matrices of elliptical distributions. Given $Z \sim EC_d(\mu, \Sigma, \xi)$, we first explore the relationship between the moments of $Z$ and $\mu$ and $\Sigma$.

**Proposition 2.5.** *Given $Z \sim EC_d(\mu, \Sigma, \xi)$ with $\mathrm{rank}(\Sigma) = q$ and finite second moments and $\Sigma^0$ the generalized correlation matrix of $Z$, we have $\mathbb{E}(Z) = \mu$, $\mathrm{Var}(Z) = \frac{\mathbb{E}(\xi^2)}{q}\Sigma$, and $\mathrm{Cor}(Z) = \Sigma^0$.*

When the random vector is elliptically distributed with second moment finite, the sample mean and correlation matrices are element-wise consistent estimators of $\mu$ and $\Sigma^0$. However, the elliptical distributions are generally very heavy-tailed (multivariate t or Cauchy distributions for example), making Pearson sample correlation matrix a bad estimator. When the distribution family is extended to the transelliptical, the Pearson sample correlation matrix is generally no longer a element-wise consistent estimator of $\Sigma^0$. A similar "plug-in" idea as [6] works when $\xi$ is known. In the general case when $\xi$ is unknown, the "plug-in" idea itself is unavailable.

## 3 The TCA

In this section we propose the TCA approach. TCA is a two-stage method in estimating the leading eigenvectors of $\Sigma^0$. Firstly, we estimate the Kendall's tau correlation matrix $\widehat{R}$. Secondly, we plug $\widehat{R}$ into a sparse PCA algorithm.

### 3.1 Rank-based Measures of Associations

The main idea of the TCA is to exploit the Kendall's tau statistic to estimate the generalized correlation matrix $\Sigma^0$ efficiently and robustly. In detail, let $X = (X_1, \ldots, X_d)^T$ be a $d-$dimensional random vector with marginal distributions $F_1, \ldots, F_d$ and the joint distributions $F_{jk}$ for the pair $(X_j, X_k)$. The population Spearman's rho and Kendall's tau correlation coefficients are given by

$$\rho(X_j, X_k) = \mathrm{Corr}(\mathrm{F_j}(X_j), \mathrm{F_k}(X_k)),$$

$$\tau(X_j, X_k) = \mathbb{P}((X_j - \widetilde{X}_j)(X_k - \widetilde{X}_k) > 0) - \mathbb{P}((X_j - \widetilde{X}_j)(X_k - \widetilde{X}_k) < 0),$$

where $(\widetilde{X}_j, \widetilde{X}_k)$ is a independent copy of $(X_j, X_k)$. In particular, for Kendall's tau, we have the following theorem, which states an explicit relationship between $\tau_{jk}$ and $\Sigma_{jk}^0$ given $X \sim TE_d(\Sigma^0, \xi; f_1, \ldots, f_d)$, no matter what the generating variable $\xi$ is. This is a strict extension to [4]'s result on the meta-elliptical distribution family.

**Theorem 3.1.** *Given $X \sim TE_d(\Sigma^0, \xi; f_1, \ldots, f_d)$ transelliptically distributed, we have*

$$\Sigma_{jk}^0 = \sin\left(\frac{\pi}{2}\tau(X_j, X_k)\right). \tag{3.1}$$

**Remark 3.1.** Although the conclusion in Theorem 3.1 of [4] is correct, the proof provided is wrong or at least very ambiguous. Theorem 2.22 in [5] builds the result only for one sample statistic and cannot be generalized to the statistic of multiple samples, like the Kendall's tau or Spearman's rho. Therefore, we provide a new and clear version here. Detailed proofs can be found in the long version of this paper [8].

Spearman's rho depends not only on $\Sigma$ but also on the generating variable $\xi$. When $X$ follows multivariate Gaussian, [17] proves that: $\rho(X_j, X_k) = \frac{6}{\pi} \arcsin(\Sigma^0_{jk}/2)$. On the other hand, when $X \sim TE_d(\Sigma^0, \xi; f_1, \ldots, f_d)$ with $\xi =^d 1$, [10] proves that: $\rho(X_j, X_k) = 3\left(\frac{\arcsin \Sigma^0_{jk}}{\pi}\right) - 4\left(\frac{\arcsin \Sigma^0_{jk}}{\pi}\right)^3$.

In estimating $\tau(X_j, X_k)$, let $x_1, \ldots, x_n$ be $n$ independent realizations of $X$, where $x_i = (x_{i1}, \ldots, x_{id})^T$. We consider the following rank-based statistic:

$$\begin{cases} \widehat{\tau}_{jk} = \dfrac{2}{n(n-1)} \displaystyle\sum_{1 \le i < i' \le n} \operatorname{sign}(x_{ij} - x_{i'j})(x_{ik} - x_{i'k}), & \text{if } j \ne k \\ \widehat{\tau}_{jk} = 1, & \text{if } j = k. \end{cases} \tag{3.2}$$

to approximate $\tau(X_j, X_k)$ and measure the association between $X_j$ and $X_k$. We define the *Kendall's tau correlation matrix* $\widehat{R} = [\widehat{R}_{jk}]$ such that $\widehat{R}_{jk} = \sin\left(\frac{\pi}{2}\widehat{\tau}_{jk}\right)$.

## 3.2 Methods

The elliptical distribution is of special interest in Principal Component Analysis (PCA). It has been shown in a variety of literatures [27, 11, 22, 12, 24] that the PCA conducted on elliptical distributions share a number of good properties enjoyed by the PCA conducted on the Gaussian distribution. We will utilize this connection to construct a new model in this paper.

### 3.2.1 TCA Model

Utilizing the natural relationship between elliptical distributions and the PCA, we propose the model of Transelliptical Component Analysis (TCA). Here ideas of transelliptical distribution family and scale-invariant PCA are exploited. We wish to estimate the leading eigenvector of the latent generalized correlation matrix. In particular, the following model $\mathcal{M}_d(\Sigma^0, \xi, s; f)$ with $f = \{f_j\}_{j=1}^d$ is considered:

$$\mathcal{M}_d(\Sigma^0, \xi, s; f): \quad \begin{cases} X \sim TE_d(\Sigma^0, \xi; f_1, \ldots, f_d), \\ \|\theta_1\|_0 = s, \end{cases} \tag{3.3}$$

where $\theta_1$ is the leading eigenvectors of the latent generalized correlation matrix $\Sigma^0$ we are interested in estimating. By spectral decomposition, we write: $\Sigma^0 = \sum_{j=1}^d \lambda_d \theta_d \theta_d^T$, where $\lambda_1 \ge \lambda_2 \ge \ldots \ge \lambda_d \ge 0$ and $\lambda_1 > 0$ to make $\Sigma^0$ non-degenerate. $\theta_1, \ldots, \theta_d \in \mathbb{S}^{d-1}$ are the corresponding eigenvectors of $\lambda_1, \ldots, \lambda_d$. Inspired by the model $\mathcal{M}_d(\Sigma^0, \xi, s; f)$, it is natural to consider the following optimization problem:

$$\widetilde{\theta}_1^* = \underset{v \in \mathbb{R}^d}{\arg\max} \; v^T \widehat{R} v,$$
$$\text{subject to} \quad v \in \mathbb{S}^{d-1} \cap \mathbb{B}_0(s), \tag{3.4}$$

where $\mathbb{B}_0(s) := \{v \in \mathbb{R}^d : \|v\|_0 \le s\}$ and $\widehat{R}$ is the estimated Kendall's tau correlation matrix. The corresponding global optimum is denoted by $\widetilde{\theta}_1^*$.

### 3.2.2 TCA Algorithm

Generally we can plug in the Kendall's tau correlation matrix $\widehat{R}$ to any sparse PCA algorithm listed above. In this paper, to approximate $\theta_1$, we consider using the Truncated Power method (TPower) proposed by [28] and [20]. The main idea of the TPower is to utilize the power method, but truncate the vector to a $\ell_0$ ball with radius $k$ in each iteration. Detailed algorithms are provided in the long version of this paper [8]. The final estimator is denoted by $\widetilde{\theta}_\infty$ with $\|\widetilde{\theta}_\infty\|_0 = k$. It will be shown in Section 4 and Section 5 that the Kendall's tau correlation matrix is a better statistic in estimating the correlation matrix than the Pearson sample correlation matrix in the sense that (i) it enjoys the Gaussian parametric rate in a much larger distribution family, including many distributions with heavy tails; (ii) it is a more robust estimator, i.e. resistant to outliers.

We use the iterative deflation method to learn the first $k$ instead of the first one leading eigenvectors, following the discussions of [21, 15, 28, 29]. In detail, a matrix $\widehat{\Gamma} \in \mathbb{R}^{d \times s}$ deflates a vector $v \in \mathbb{R}^d$ and achieves a new matrix $\widehat{\Gamma}'$: $\widehat{\Gamma}' := (I - vv^T)\widehat{\Gamma}(I - vv^T)$. In this way, $\widehat{\Gamma}'$ is orthogonal to $v$.

# 4 Theoretical Properties

In this section the theoretical properties of the TCA estimators are provided. Especially, we are interested in the high dimensional case when $d > n$.

## 4.1 Rank-based Correlation Matrix Estimation

This section is devoted to the concentration result of the Kendall sample correlation matrix $\widehat{R}$ to the Pearson correlation matrix $\Sigma^0$. The $\ell_{\max}$ convergence rate of $\widehat{R}$ is provided in the next theorem.

**Theorem 4.1.** *Given $x_1, \ldots, x_n$ $n$ independent realizations of $X \sim TE_d(\Sigma^0, \xi; f_1, \ldots, f_d)$ and letting $\widehat{R}$ be the Kendall tau correlation matrix, we have with probability at least $1 - d^{-5/2}$,*

$$\|\widehat{R} - \Sigma^0\|_{\max} \le 3\pi \sqrt{\log d / n}. \tag{4.1}$$

*Proof sketch.* Theorem 4.1 can be proved by realizing that $\widehat{\tau}_{jk}$ is an unbiased estimator of $\tau(X_j, X_k)$ and is a U-statistic with size 2. Hoeffding's inequality for U-statistic can then be applied to obtain the result. Detailed proofs can be found in the long version of this paper [8]. $\qquad\square$

## 4.2 TCA Estimators

This section is devoted to the statement of our main result on the upper bound of the estimated error of the TCA global optimum $\widetilde{\theta}_1^*$ and TPower solver $\widetilde{\theta}_\infty$. We assume that the Model $\mathcal{M}_d(\Sigma^0, \xi, s; f)$ holds and the next theorem provides an upper bound on the angle between the estimated leading eigenvector $\widetilde{\theta}_1^*$ and true leading eigenvector $\theta_1$.

**Theorem 4.2.** *Let $\widetilde{\theta}_1^*$ be the global solution to Equation* (3.4) *and the Model $\mathcal{M}_d(\Sigma^0, \xi, s; f)$ holds. For any two vectors $v_1 \in \mathbb{S}^{d-1}$ and $v_2 \in \mathbb{S}^{d-1}$, letting*

$$|\sin\angle(v_1, v_2)| = \sqrt{1 - (v_1^T v_2)^2},$$

*then we have*

$$\mathbb{P}\left(|\sin\angle(\widetilde{\theta}_1^*, \theta_1)| \le \frac{6\pi}{\lambda_1 - \lambda_2} \cdot s\sqrt{\frac{\log d}{n}}\right) \ge 1 - d^{-5/2}. \tag{4.2}$$

*Proof sketch.* The key idea of the proof is to utilize the $\ell_{\max}$ norm convergence result of $\widehat{R}$ to $\Sigma^0$. Detailed proofs can be found in the long version of this paper [8]. $\qquad\square$

Generally, when $s$ and $\lambda_1, \lambda_2$ do not scale with $(n, d)$, the rate is $O_P(\sqrt{\log d / n})$, which is the parametric rate [20, 26, 23] obtains. When $(n, d)$ goes to infinity, the two leading eigenvalues $\lambda_1$ and $\lambda_2$ will typically go to infinity and will at least be away from zero. Hence, our rate shown in Theorem 4.2 will be usually better than the seemingly more common rate: $\frac{6\pi\lambda_1}{\lambda_1 - \lambda_2} \cdot s\sqrt{\frac{\log d}{n}}$.

**Corollary 4.1 (Feature Selection Consistency of the TCA).** *Let $\widetilde{\theta}_1^*$ be the global solution to Equation* (3.4) *and the Model $\mathcal{M}_d(\Sigma^0, \xi, s; f)$ holds. Let*

$$\Theta := supp(\theta_1) \quad \text{and} \quad \widehat{\Theta}^* := supp(\widetilde{\theta}_1^*).$$

*If we further have*

$$\min_{j \in \Theta} |\theta_{1j}| \ge \frac{6\sqrt{2}\pi}{\lambda_1 - \lambda_2} \cdot s\sqrt{\frac{\log d}{n}},$$

*then we have, $\mathbb{P}(\widehat{\Theta}^* = \Theta) \ge 1 - d^{-5/2}$.*

*Proof sketch.* The key of the proof is to construct a contradiction given Theorem 4.2 and the condition on the minimum value of $|\theta_1|$. Detailed proofs can be found in the long version of this paper [8]. $\qquad\square$

# 5 Experiments

In this section we investigate the empirical performance of the TCA method. We utilize the TPower algorithm proposed by [28] and the following three methods are considered: (1) Pearson: the classic high dimensional scale-invariant PCA using the Pearson sample correlation matrix of the data; (2) Kendall: the TCA using the Kendall correlation matrix; (3) LatPearson: the classic high dimensional scale-invariant PCA using the Pearson sample correlation matrix of the data drawn from the latent elliptical distribution (perfect without data contamination).

## 5.1 Numerical Simulations

In the simulation study we randomly sample $n$ data points from a certain transelliptical distribution $TE_d(\Sigma^0, \xi; f_1, \ldots, f_d)$. Here we consider the set up of $d = 100$. To determine the transelliptical distribution, firstly, we derive $\Sigma^0$ in the following way: A covariance matrix $\Sigma$ is firstly synthesized through the eigenvalue decomposition, where the first two eigenvalues are given and the corresponding eigenvectors are pre-specified to be sparse. In detail, let $\Sigma = \sum_{j=1}^{d} \omega_j u_j u_j^T$, where $\omega_1 = 6, \omega_2 = 3, \omega_3 = \ldots = \omega_d = 1$, and the first two leading eigenvectors of $\Sigma$, $u_1$ and $u_2$, are sparse with the first $s = 10$ entries of $u_1$ and the second $s = 10$ entries of $u_2$ are nonzero, i.e.

$$u_{1j} = \left\{ \begin{array}{ll} \frac{1}{\sqrt{10}} & 1 \leq j \leq 10 \\ 0 & \text{otherwise} \end{array} \right. \quad \text{and} \quad u_{2j} = \left\{ \begin{array}{ll} \frac{1}{\sqrt{10}} & 11 \leq j \leq 20 \\ 0 & \text{otherwise} \end{array} \right. . \tag{5.1}$$

The remaining eigenvectors are chosen arbitrarily. The generalized correlation matrix $\Sigma^0$ is generated from $\Sigma$, with $\lambda_1 = 4, \lambda_2 = 2.5, \lambda_3, \ldots, \lambda_d \leq 1$ and the top two leading eigenvectors sparse:

$$\theta_{1j} = \left\{ \begin{array}{ll} -\frac{1}{\sqrt{10}} & 1 \leq j \leq 10 \\ 0 & \text{otherwise} \end{array} \right. \quad \text{and} \quad \theta_{2j} = \left\{ \begin{array}{ll} -\frac{1}{\sqrt{10}} & 11 \leq j \leq 20 \\ 0 & \text{otherwise} \end{array} \right. . \tag{5.2}$$

Secondly, using $\Sigma^0$, we consider the following three generating schemes:

**[Scheme 1]** $X \sim TE_d(\Sigma^0, \xi; f_1, \ldots, f_d)$ with $\xi \sim \chi_d$ and $f_1(x) = \ldots = f_d(x) = x$. Here $\sqrt{Y_1^2 + \ldots + Y_d^2} \sim \chi_d$ with $Y_1, \ldots, Y_d \sim^{i.i.d} N(0,1)$. In other words, $\chi_d$ is the chi-distribution with degree of freedom $d$. This is equivalent to say that $X \sim N(0, \Sigma^0)$ (Example 2.4 of [5]).

**[Scheme 2]** $X \sim TE_d(\Sigma^0, \xi; f_1, \ldots, f_d)$ with $\xi =^d \sqrt{m}\xi_1^*/\xi_2^*$ and $f_1(x) = \ldots = f_d(x) = x$. Here $\xi_1^* \sim \chi_d$, $\xi_2^* \sim \chi_m$, $\xi_1^*$ is independent of $\xi_2^*$ and $m \in \mathbb{N}$. This is equivalent to say that $X \sim Mt_d(m, \mathbf{0}, \Sigma^0)$, i.e. $X$ following a multivariate-t distribution with degree of freedom $m$, mean $\mathbf{0}$ and covariance matrix $\Sigma^0$ (Example 2.5 of [5]). Here we consider $m = 3$.

**[Scheme 3]** $X \sim TE_d(\Sigma^0, \xi; f_1, \ldots, f_d)$ with $\xi =^d \sqrt{m}\xi_1^*/\xi_2^*$. Here $\xi_1^* \sim \chi_d$, $\xi_2^* \sim \chi_m$, $\xi_1^*$ is independent of $\xi_2^*$ and $m = 3$. Moreover, $\{f_1, \ldots, f_d\} = \{h_1, h_2, h_3, h_4, h_5, h_1, h_2, h_3, h_4, h_5, \ldots\}$, where

$$h_1^{-1}(x) := x, \quad h_2^{-1}(x) := \frac{\text{sign}(x)|x|^{1/2}}{\sqrt{\int |t|\phi(t)dt}}, h_3^{-1}(x) := \frac{\Phi(x) - \int \Phi(t)\phi(t)dt}{\sqrt{\int \left(\Phi(y) - \int \Phi(t)\phi(t)dt\right)^2 \phi(y)dy}},$$

$$h_4^{-1}(x) := \frac{x^3}{\sqrt{\int t^6\phi(t)dt}}, h_5^{-1}(x) := \frac{\exp(x) - \int \exp(t)\phi(t)dt}{\sqrt{\int \left(\exp(y) - \int \exp(t)\phi(t)dt\right)^2 \phi(y)dy}}.$$

This is equivalent to say that $X$ is transelliptically distributed with the latent elliptical distribution $Z \sim Mt_d(3, \mathbf{0}, \Sigma^0)$.

To evaluate the robustness of different methods, let $r \in [0, 1)$ represent the proportion of samples being contaminated. For each dimension, we randomly select $\lfloor nr \rfloor$ entries and replace them with either 5 or -5 with equal probability. The final data matrix we obtained is $\boldsymbol{X} \in \mathbb{R}^{n \times d}$. Here we pick $r = 0, 0.02$ or $0.05$. Under the Scheme 1 to Scheme 3 with different levels of contamination ($r = 0, 0.02$ or $0.05$), we repeatedly generate the data matrix $\boldsymbol{X}$ for 1,000 times and compute the averaged False Positive Rates and False Negative Rates using a path of tuning parameters $k$ from 5 to 90. The feature selection performances of different methods are then evaluated by plotting $(\text{FPR}(k), 1 - \text{FNR}(k))$. The corresponding ROC curves are presented in Figure 1 (A). More results are shown in the long version of this paper [8]. It can be observed that Kendall is generally better and more resistance to the outliers compared with Pearson.

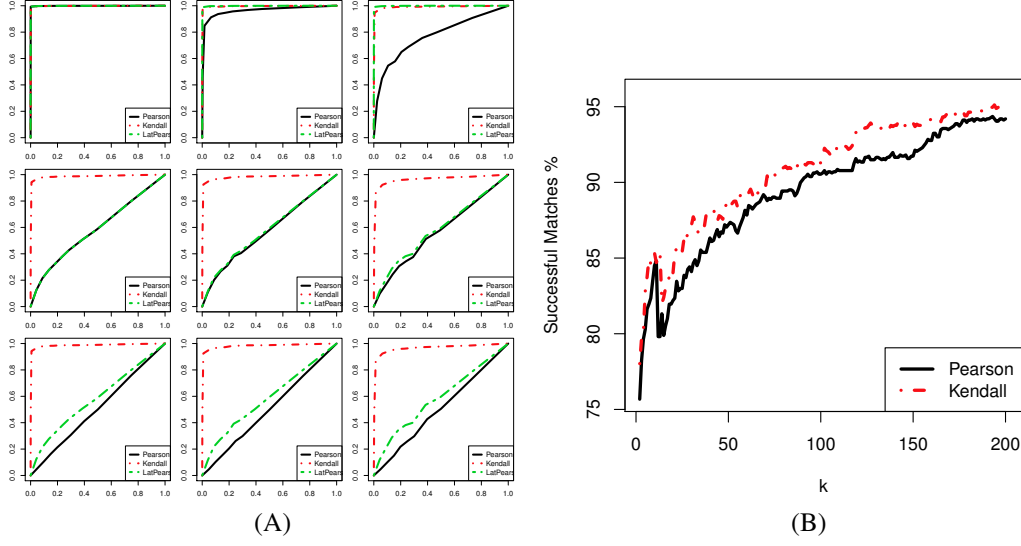

Figure 1: (A) ROC curves under Scheme 1, Scheme 2 and Scheme 3 (top, middle, bottom) and data contamination at different levels ($r = 0, 0.02, 0.05$ from left to right). $x-$axis is FPR and $y-$axis is TPR. Here $n = 100$ and $d = 100$. (B) Successful matches of the market trend proportions only using the stocks in $A_k$ and $B_k$. The $x-$axis represents the tuning parameter $k$ scaling from 1 to 200; the $y-$axis represents the % of successful matches. The curve denoted by 'Kendall' represents the points of $(k, \rho_{A_k})$ and the curves denoted by 'Pearson' represents the points of $(k, \rho_{B_k})$.

## 5.2   Equities Data

In this section we apply the TCA on the stock price data from Yahoo! Finance (`finance.yahoo.com`). We collected the daily closing prices for J=452 stocks that were consistently in the S&P 500 index between January 1, 2003 through January 1, 2008. This gave us altogether T=1,257 data points, each data point corresponds to the vector of closing prices on a trading day. Let $St = [St_{t,j}]$ denote by the closing price of stock $j$ on day $t$.

We wish to evaluate the ability of using the only $k$ stocks to represent the trend of the whole stock market. To this end, we run Kendall and Pearson on $St$ and obtain the leading eigenvectors $\widetilde{\theta}_{Kendall}$ and $\widetilde{\theta}_{Pearson}$ using the tuning parameter $k \in \mathbb{N}$. Let $A_k := \mathrm{supp}(\widetilde{\theta}_{Kendall})$ and $B_k := \mathrm{supp}(\widetilde{\theta}_{Pearson})$. And then we let $T_t^W$, $T_t^{A_k}$ and $T_t^{B_k}$ denote by the trend of the whole stocks, $A_k$ stocks and $B_k$ stocks in $t$th day compared with $t - 1$th date, i.e:

$$T_t^W := I(\sum_j St_{t,j} - \sum_j St_{t-1,j} >), T_t^{A_k} := I(\sum_{j \in A_k} St_{t,j} - \sum_{j \in A_k} St_{t-1,j} > 0)$$

and

$$T_t^{B_k} := I(\sum_{j \in B_k} St_{t,j} - \sum_{j \in B_k} St_{t-1,j} > 0),$$

here $I$ is the indicator function. In this way, we can calculate the proportion of successful matches of the market trend using the stocks in $A_k$ and $B_k$ as: $\rho_{A_k} := \frac{1}{T} \sum_t I(T_t^W = T_t^{A_k})$ and $\rho_{B_k} := \frac{1}{T} \sum_t I(T_t^W = T_t^{B_k})$. We visualize the result by plotting $(k, \rho_{A_k})$ and $(k, \rho_{B_k})$ on a 2D figure. The result is presented in Figure 1 (B).

It can be observed from Figure 1 (B) that Kendall summarizes the trend of the whole stock market constantly better than Pearson. Moreover, the averaged difference between the two methods are $\frac{1}{200} \sum_k (\rho_{A_k} - \rho_{B_k}) = 1.4025$ with the standard deviation 0.6743. Therefore, the difference is significant.

## 6   Acknowledgement

This research was supported by NSF award IIS-1116730.

# References

[1] TW Anderson. Statistical inference in elliptically contoured and related distributions. *Recherche*, 67:02, 1990.

[2] M.G. Borgognone, J. Bussi, and G. Hough. Principal component analysis in sensory analysis: covariance or correlation matrix? *Food quality and preference*, 12(5-7):323–326, 2001.

[3] S. Cambanis, S. Huang, and G. Simons. On the theory of elliptically contoured distributions. *Journal of Multivariate Analysis*, 11(3):368–385, 1981.

[4] H.B. Fang, K.T. Fang, and S. Kotz. The meta-elliptical distributions with given marginals. *Journal of Multivariate Analysis*, 82(1):1–16, 2002.

[5] KT Fang, S. Kotz, and KW Ng. Symmetric multivariate and related distributions. *Chapman&Hall, London*, 1990.

[6] C. Genest, AC Favre, J. Béliveau, and C. Jacques. Metaelliptical copulas and their use in frequency analysis of multivariate hydrological data. *Water Resour. Res*, 43(9):W09401, 2007.

[7] P.R. Halmos. *Measure theory*, volume 18. Springer, 1974.

[8] F. Han and H. Liu. Tca: Transelliptical principal component analysis for high dimensional non-gaussian data. *Technical Report*, 2012.

[9] W. Hoeffding. Probability inequalities for sums of bounded random variables. *Journal of the American Statistical Association*, pages 13–30, 1963.

[10] H. Hult and F. Lindskog. Multivariate extremes, aggregation and dependence in elliptical distributions. *Advances in Applied probability*, 34(3):587–608, 2002.

[11] DR Jensen. The structure of ellipsoidal distributions, ii. principal components. *Biometrical Journal*, 28(3):363–369, 1986.

[12] DR Jensen. Conditioning and concentration of principal components. *Australian Journal of Statistics*, 39(1):93–104, 1997.

[13] H. Joe. *Multivariate models and dependence concepts*, volume 73. Chapman & Hall/CRC, 1997.

[14] I.M. Johnstone and A.Y. Lu. Sparse principal components analysis. *Arxiv preprint arXiv:0901.4392*, 2009.

[15] M. Journée, Y. Nesterov, P. Richtárik, and R. Sepulchre. Generalized power method for sparse principal component analysis. *The Journal of Machine Learning Research*, 11:517–553, 2010.

[16] KS Kelly and R. Krzysztofowicz. A bivariate meta-gaussian density for use in hydrology. *Stochastic Hydrology and Hydraulics*, 11(1):17–31, 1997.

[17] W.H. Kruskal. Ordinal measures of association. *Journal of the American Statistical Association*, pages 814–861, 1958.

[18] D. Kurowicka, J. Misiewicz, and RM Cooke. Elliptical copulae. In *Proc of the International Conference on Monte Carlo Simulation-Monte Carlo*, pages 209–214, 2000.

[19] H. Liu, J. Lafferty, and L. Wasserman. The nonparanormal: Semiparametric estimation of high dimensional undirected graphs. *The Journal of Machine Learning Research*, 10:2295–2328, 2009.

[20] Z. Ma. Sparse principal component analysis and iterative thresholding. *Arxiv preprint arXiv:1112.2432*, 2011.

[21] L. Mackey. Deflation methods for sparse pca. *Advances in neural information processing systems*, 21:1017–1024, 2009.

[22] G.P. McCabe. Principal variables. *Technometrics*, pages 137–144, 1984.

[23] D. Paul and I.M. Johnstone. Augmented sparse principal component analysis for high dimensional data. *Arxiv preprint arXiv:1202.1242*, 2012.

[24] GQ Qian, G. Gabor, and RP Gupta. Principal components selection by the criterion of the minimum mean difference of complexity. *Journal of multivariate analysis*, 49(1):55–75, 1994.

[25] A. Sklar. Fonctions de répartition à n dimensions et leurs marges. *Publ. Inst. Statist. Univ. Paris*, 8(1):11, 1959.

[26] V.Q. Vu and J. Lei. Minimax rates of estimation for sparse pca in high dimensions. *Arxiv preprint arXiv:1202.0786*, 2012.

[27] C.M. Waternaux. Principal components in the nonnormal case: The test of equality of q roots. *Journal of Multivariate Analysis*, 14(3):323–335, 1984.

[28] X.T. Yuan and T. Zhang. Truncated power method for sparse eigenvalue problems. *Arxiv preprint arXiv:1112.2679*, 2011.

[29] Y. Zhang, A. dAspremont, and L.E. Ghaoui. Sparse pca: Convex relaxations, algorithms and applications. *Handbook on Semidefinite, Conic and Polynomial Optimization*, pages 915–940, 2012.

